# Bayesian Action-Graph Games

**Albert Xin Jiang**
Department of Computer Science
University of British Columbia
jiang@cs.ubc.ca

**Kevin Leyton-Brown**
Department of Computer Science
University of British Columbia
kevinlb@cs.ubc.ca

## Abstract

Games of incomplete information, or Bayesian games, are an important game-theoretic model and have many applications in economics. We propose Bayesian action-graph games (BAGGs), a novel graphical representation for Bayesian games. BAGGs can represent arbitrary Bayesian games, and furthermore can compactly express Bayesian games exhibiting commonly encountered types of structure including symmetry, action- and type-specific utility independence, and probabilistic independence of type distributions. We provide an algorithm for computing expected utility in BAGGs, and discuss conditions under which the algorithm runs in polynomial time. Bayes-Nash equilibria of BAGGs can be computed by adapting existing algorithms for complete-information normal form games and leveraging our expected utility algorithm. We show both theoretically and empirically that our approaches improve significantly on the state of the art.

## 1 Introduction

In the last decade, there has been much research at the interface of computer science and game theory (see e.g. [19, 22]). One fundamental class of computational problems in game theory is the computation of *solution concepts* of a finite game. Much of current research on computation of solution concepts has focused on *complete-information games*, in which the game being played is common knowledge among the players. However, in many multi-agent situations, players are uncertain about the game being played. Harsanyi [10] proposed games of incomplete information (or Bayesian games) as a mathematical model of such interactions. Bayesian games have found many applications in economics, including most notably auction theory and mechanism design.

Our interest is in computing with Bayesian games, and particularly in identifying sample Bayes-Nash equilibrium. There are two key obstacles to performing such computations efficiently. The first is representational: the straightforward tabular representation of Bayesian game utility functions (the Bayesian Normal Form) requires space exponential in the number of players. For large games, it becomes infeasible to store the game in memory, and performing even computations that are polynomial time in the input size are impractical. An analogous obstacle arises in the context of complete-information games: there the standard representation (normal form) also requires space exponential in the number of players. The second obstacle is the lack of existing algorithms for identifying sample Bayes-Nash equilibrium for arbitrary Bayesian games. Harsanyi [10] showed that a Bayesian game can be interpreted as an equivalent complete-information game via "induced normal form" or "agent form" interpretations. Thus one approach is to interpret a Bayesian game as a complete-information game, enabling the use of existing Nash-equilibrium-finding algorithms (e.g. [24, 9]). However, generating the normal form representations under both of these complete-information interpretations causes a *further* exponential blowup in representation size.

Most games of interest have highly-structured payoff functions, and thus it is possible to overcome the first obstacle by representing them compactly. This has been done for complete information games through (e.g.) the graphical games [16] and Action-Graph Games (AGGs) [1] representations. In this paper we propose Bayesian Action-Graph Games (BAGGs), a compact representation for

Bayesian games. BAGGs can represent arbitrary Bayesian games, and furthermore can compactly express Bayesian games with commonly encountered types of structure. The type profile distribution is represented as a Bayesian network, which can exploit conditional independence structure among the types. BAGGs represent utility functions in a way similar to the AGG representation, and like AGGs, are able to exploit anonymity and action-specific utility independencies. Furthermore, BAGGs can compactly express Bayesian games exhibiting *type-specific independence*: each player's utility function can have different kinds of structure depending on her instantiated type. We provide an algorithm for computing expected utility in BAGGs, a key step in many algorithms for game-theoretic solution concepts. Our approach interprets expected utility computation as a probabilistic inference problem on an *induced Bayesian Network*. In particular, our algorithm runs in polynomial time for the important case of independent type distributions. To compute Bayes-Nash equilibria for BAGGs, we consider the agent form interpretation of the BAGG. Although a naive normal form representation would require an exponential blowup, BAGGs can act as a compact representation of the agent form. Computational tasks on the agent form can be done efficiently by leveraging our expected utility algorithm for BAGGs. We have implemented our approach by adapting two Nash equilibrium algorithms, the simplicial subdivision algorithm [24] and Govindan and Wilson's global Newton method [9]. We show empirically that our approach outperforms the existing approaches of solving for Nash on the induced normal form or on the normal form representation of the agent form.

We now discuss some related literature. There has been some research on heuristic methods for finding Bayes-Nash equilibria for certain classes of auction games using iterated best response (see e.g. [21, 25]). Such methods are not guaranteed to converge to a solution. Howson and Rosenthal [12] applied the agent form transformation to 2-player Bayesian games, resulting in a complete-information polymatrix game. Our approach can be seen as a generalization of their method to general Bayesian games. Singh et al. [23] proposed a incomplete information version of the graphical game representation, and presented efficient algorithms for computing approximate Bayes-Nash equilibria in the case of tree games. Gottlob et al. [7] considered a similar extension of the graphical game representation and analyzed the problem of finding a pure-strategy Bayes-Nash equilibrium. Like graphical games, such representations are limited in that they can only exploit *strict* utility independencies. Oliehoek et al. [20] proposed a heuristic search algorithm for common-payoff Bayesian games, which has applications to cooperative multi-agent problems. Bayesian games can be interpreted as dynamic games with a initial move by Nature; thus, also related is the literature on representations for dynamic games, including multi-agent influence diagrams (MAIDs) [17] and temporal action-graph games (TAGGs) [14]. Compared to these representations for dynamic games, BAGGs focus explicitly on structure common to Bayesian games; in particular, only BAGGs can efficiently express type-specific utility structure. Also, by representing utility functions and type distributions as separate components, BAGGs can be more versatile (e.g., a future direction is to answer computational questions that do not depend on the type distribution, such as ex-post equilibria). Furthermore, BAGGs can be solved by adapting Nash-equilibrium algorithms such as Govindan and Wilson's global Newton method [9] for static games; this is generally more practical than their related Nash equilibrium algorithm [8] that directly works on dynamic games: while both approach avoids the exponential blowup of transforming to the induced normal form, the algorithm for dynamic games has to solve an additional quadratic program at each step.

## 2 Preliminaries

### 2.1 Complete-information Games

We assume readers are familiar with the basic concepts of complete-information games and here we only establish essential notation. A *complete-information game* is a tuple $(N, \{A_i\}_{i \in N}, \{u_i\}_{i \in N})$ where $N = \{1, \ldots, n\}$ is the set of agents; for each agent $i$, $A_i$ is the set of $i$'s actions. We denote by $a_i \in A_i$ one of $i$'s actions. An action profile $a = (a_1, \ldots, a_n) \in \prod_{i \in N} A_i$ is a tuple of the agents' actions. Agent $i$'s utility function is $u_i : \prod_{j \in N} A_j \to \mathbb{R}$. A *mixed strategy* $\sigma_i$ for player $i$ is a probability distribution over $A_i$. A mixed strategy profile $\sigma$ is a tuple of the $n$ players' mixed strategies. We denote by $u_i(\sigma)$ the expected utility of player $i$ under the mixed strategy profile $\sigma$. We adopt the following notational convention: for any $n$-tuple $X$ we denote by $X_{-i}$ the elements of $X$ corresponding to players other than $i$.

A *game representation* is a data structure that stores all information needed to specify a game. A *normal form* representation of a game uses a matrix to represent each utility function $u_i$. The size of this representation is $n \prod_{j \in N} |A_j|$, which grows exponentially in the number of players.

## 2.2 Bayesian Games

We now define Bayesian games and discuss common types of structure.

**Definition 1.** *A Bayesian game is a tuple* $(N, \{A_i\}_{i \in N}, \Theta, P, \{u_i\}_{i \in N})$ *where* $N = \{1, \ldots, n\}$ *is the set of players; each* $A_i$ *is player* $i$'s *action set, and* $A = \prod_i A_i$ *is the set of action profiles;* $\Theta = \prod_i \Theta_i$ *is the set of type profiles, where* $\Theta_i$ *is player* $i$'s *set of types;* $P : \Theta \rightarrow \mathbb{R}$ *is the type distribution and* $u_i : A \times \Theta \rightarrow \mathbb{R}$ *is the utility function for player* $i$.

As in the complete-information case, we denote by $a_i$ an element of $A_i$, and $a = (a_1, \ldots, a_n)$ an action profile. Furthermore we denote by $\theta_i$ an element of $\Theta_i$, and by $\theta$ a type profile. The game is played as follows. A type profile $\theta = (\theta_1, \ldots, \theta_n) \in \Theta$ is drawn according to the distribution $P$. Each player $i$ observes her type $\theta_i$ and, based on this observation, chooses from her set of actions $A_i$. Each player $i$'s utility is then given by $u_i(a, \theta)$, where $a$ is the resulting action profile.

Player $i$ can deterministically choose a *pure strategy* $s_i$, in which given each $\theta_i \in \Theta_i$ she deterministically chooses an action $s_i(\theta_i)$. Player $i$ can also randomize and play a *mixed strategy* $\sigma_i$, in which her probability of choosing $a_i$ given $\theta_i$ is $\sigma_i(a_i | \theta_i)$. That is, given a type $\theta_i \in \Theta_i$, she plays according to distribution $\sigma_i(\cdot | \theta_i)$ over her set of actions $A_i$. A mixed strategy profile $\sigma = (\sigma_1, \ldots, \sigma_n)$ is a tuple of the players' mixed strategies.

The *expected utility* of $i$ given $\theta_i$ under a mixed strategy profile $\sigma$ is the expected value of $i$'s utility under the resulting joint distribution of $a$ and $\theta$, conditioned on $i$ receiving type $\theta_i$:

$$u_i(\sigma | \theta_i) = \sum_{\theta_{-i}} P(\theta_{-i} | \theta_i) \sum_a u_i(a, \theta) \prod_j \sigma_j(a_j | \theta_j). \tag{1}$$

A mixed strategy profile $\sigma$ is a *Bayes-Nash equilibrium* if for all $i$, for all $\theta_i$, for all $a_i \in A_i$, $u_i(\sigma | \theta_i) \geq u_i(\sigma^{\theta_i \rightarrow a_i} | \theta_i)$, where $\sigma^{\theta_i \rightarrow a_i}$ is the mixed strategy profile that is identical to $\sigma$ except that $i$ plays $a_i$ with probability 1 given $\theta_i$.

In specifying a Bayesian game, the space bottlenecks are the type distribution and the utility functions. Without additional structure, we cannot do better than representing each utility function $u_i : A \times \Theta \rightarrow R$ as a table and the type distribution as a table as well. We call this representation the *Bayesian normal form*. The size of this representation is $n \times \prod_{i=1}^n (|\Theta_i| \times |A_i|) + \prod_{i=1}^n |\Theta_i|$.

We say a Bayesian game has *independent type distributions* if players' types are drawn independently, i.e. the type-profile distribution $P(\theta)$ is a product distribution: $P(\theta) = \prod_i P(\theta_i)$. In this case the distribution $P$ can be represented compactly using $\sum_i |\Theta_i|$ numbers.

Given a permutation of players $\pi : N \rightarrow N$ and an action profile $a = (a_1, \ldots, a_n)$, let $a^\pi = (a_{\pi(1)}, \ldots, a_{\pi(n)})$. Similarly let $\theta^\pi = (\theta_{\pi(1)}, \ldots, \theta_{\pi(n)})$. We say the type distribution $P$ is symmetric if $|\Theta_i| = |\Theta_j|$ for all $i, j \in N$, and if for all permutations $\pi : N \rightarrow N$, $P(\theta) = P(\theta^\pi)$. We say a Bayesian game has *symmetric utility functions* if $|A_i| = |A_j|$ and $|\Theta_i| = |\Theta_j|$ for all $i, j \in N$, and if for all permutations $\pi : N \rightarrow N$, we have $u_i(a, \theta) = u_{\pi(i)}(a^\pi, \theta^\pi)$ for all $i \in N$. A Bayesian game is symmetric if its type distribution and utility functions are symmetric. The utility functions of such a game range over at most $|\Theta_i||A_i| \binom{n-2+|\Theta_i||A_i|}{|\Theta_i||A_i|-1}$ unique utility values.

A Bayesian game exhibits *conditional utility independence* if each player $i$'s utility depends on the action profile $a$ and her own type $\theta_i$, but does not depend on the other players' types. Then the utility function of each player $i$ ranges over at most $|A||\Theta_i|$ unique utility values.

### 2.2.1 Complete-information interpretations

Harsanyi [10] showed that any Bayesian game can be interpreted as a complete-information game, such that Bayes-Nash equilibria of the Bayesian game correspond to Nash equilibria of the complete-information game. There are two complete-information interpretations of Bayesian games.

A Bayesian game can be converted to its *induced normal form*, which is a complete-information game with the same set of $n$ players, in which each player's set of actions is her set of pure strategies in the Bayesian game. Each player's utility under an action profile is defined to be equal to the player's expected utility under the corresponding pure strategy profile in the Bayesian game.

Alternatively, a Bayesian game can be transformed to its *agent form*, where each type of each player in the Bayesian game is turned into one player in a complete-information game. Formally, given a

Bayesian game $(N, \{A_i\}_{i \in N}, \Theta, P, \{u_i\}_{i \in N})$, we define its agent form as the complete-information game $(\tilde{N}, \{\tilde{A}_{j,\theta_j}\}_{(j,\theta_j) \in \tilde{N}}, \{\tilde{u}_{j,\theta_j}\}_{(j,\theta_j) \in \tilde{N}})$, where $\tilde{N}$ consists of $\sum_{j \in N} |\Theta_j|$ players, one for every type of every player of the Bayesian game. We index the players by the tuple $(j, \theta_j)$ where $j \in N$ and $\theta_j \in \Theta_j$. For each player $(j, \theta_j) \in \tilde{N}$ of the agent form game, her action set $\tilde{A}_{(j,\theta_j)}$ is $A_j$, the action set of $j$ in the Bayesian game. The set of action profiles is then $\tilde{A} = \prod_{j,\theta_j} A_{(j,\theta_j)}$. The utility function of player $(j, \theta_j)$ is $\tilde{u}_{j,\theta_j} : \tilde{A} \to \mathbb{R}$. For all $\tilde{a} \in \tilde{A}$, $\tilde{u}_{j,\theta_j}(\tilde{a})$ is equal to the expected utility of player $j$ of the Bayesian game given type $\theta_j$, under the pure strategy profile $s^{\tilde{a}}$, where for all $i$ and all $\theta_i$, $s_i^{\tilde{a}}(\theta_i) = \tilde{a}_{(i,\theta_i)}$. Observe that there is a one-to-one correspondence between action profiles in the agent form and pure strategies of the Bayesian game. A similar correspondence exists for mixed strategy profiles: each mixed strategy profile $\sigma$ of the Bayesian game corresponds to a mixed strategy $\tilde{\sigma}$ of the agent form, with $\tilde{\sigma}_{(i,\theta_i)}(a_i) = \sigma_i(a_i|\theta_i)$ for all $i, \theta_i, a_i$. It is straightforward to verify that $\tilde{u}_{i,\theta_i}(\tilde{\sigma}) = u_i(\sigma|\theta_i)$ for all $i, \theta_i$. This implies a correspondence between Bayes Nash equilibria of a Bayesian game and Nash equilibria of its agent form.

**Proposition 2.** *$\sigma$ is a Bayes-Nash equilibrium of a Bayesian game if and only if $\tilde{\sigma}$ is a Nash equilibrium of its agent form.*

## 3 Bayesian Action-Graph Games

In this section we introduce Bayesian Action-Graph Games (BAGGs), a compact representation of Bayesian games. First consider representing the type distributions. Specifically, the type distribution $P$ is specified by a Bayesian network (BN) containing at least $n$ random variables corresponding to the $n$ players' types $\theta_1, \ldots, \theta_n$. For example, when the types are independently distributed, then $P$ can be specified by the simple BN with $n$ variables $\theta_1, \ldots, \theta_n$ and no edges.

Now consider representing the utility functions. Our approach is to adapt concepts from the AGG representation [1, 13] to the Bayesian game setting. At a high level, a BAGG is a Bayesian game on an *action graph*, a directed graph on a set of *action nodes* $\mathcal{A}$. To play the game, each player $i$, given her type $\theta_i$, simultaneously chooses an action node from her *type-action set* $A_{i,\theta_i} \subseteq \mathcal{A}$. Each action node thus corresponds to an action choice that is available to one or more of the players. Once the players have made their choices, an *action count* is tallied for each action node $\alpha \in \mathcal{A}$, which is the number of agents that have chosen $\alpha$. A player's utility depends only on the action node she chose and the action counts on the neighbors of the chosen node.

We now turn to a formal description of BAGG's utility function representation. Central to our model is the *action graph*. An *action graph* $G = (\mathcal{A}, E)$ is a directed graph where $\mathcal{A}$ is the set of action nodes, and $E$ is a set of directed edges, with self edges allowed. We say $\alpha'$ is a *neighbor* of $\alpha$ if there is an edge from $\alpha'$ to $\alpha$, i.e., if $(\alpha', \alpha) \in E$. Let the *neighborhood* of $\alpha$, denoted $\nu(\alpha)$, be the set of neighbors of $\alpha$.

For each player $i$ and each instantiation of her type $\theta_i \in \Theta_i$, her *type-action set* $A_{i,\theta_i} \subseteq \mathcal{A}$ is the set of possible action choices of $i$ given $\theta_i$. These subsets are unrestricted: different type-action sets may (partially or completely) overlap. Define player $i$'s *total action set* to be $A_i^{\cup} = \bigcup_{\theta_i \in \Theta_i} A_{i,\theta_i}$. We denote by $A = \prod_i A_i^{\cup}$ the set of *action profiles*, and by $a \in A$ an action profile. Observe that the action profile $a$ provides sufficient information about the type profile to be able to determine the outcome of the game; there is no need to additionally encode the realized type distribution. We note that for different types $\theta_i, \theta_i' \in \Theta_i$, $A_{i,\theta_i}$ and $A_{i,\theta_i'}$ may have different sizes; i.e., $i$ may have different numbers of available action choices depending on her realized type.

A *configuration* $c$ is a vector of $|\mathcal{A}|$ non-negative integers, specifying for each action node the numbers of players choosing that action. Let $c(\alpha)$ be the element of $c$ corresponding to the action $\alpha$. Let $\mathcal{C} : A \mapsto C$ be the function that maps from an action profile $a$ to the corresponding configuration $c$. Formally, if $c = \mathcal{C}(a)$ then $c(\alpha) = |\{i \in N : a_i = \alpha\}|$ for all $\alpha \in \mathcal{A}$. Define $C = \{c : \exists a \in A \text{ such that } c = \mathcal{C}(a)\}$. In other words, $C$ is the set of all possible configurations. We can also define a configuration over a subset of nodes. In particular, we will be interested in configurations over a node's neighborhood. Given a configuration $c \in C$ and a node $\alpha \in \mathcal{A}$, let the *configuration over the neighborhood* of $\alpha$, denoted $c^{(\alpha)}$, be the restriction of $c$ to $\nu(\alpha)$, i.e., $c^{(\alpha)} = (c(\alpha'))_{\alpha' \in \nu(\alpha)}$. Similarly, let $C^{(\alpha)}$ denote the set of configurations over $\nu(\alpha)$ in which at least one player plays $\alpha$. Let $\mathcal{C}^{(\alpha)} : A \mapsto C^{(\alpha)}$ be the function which maps from an action profile to the corresponding configuration over $\nu(\alpha)$.

**Definition 3.** *A Bayesian action-graph game (BAGG) is a tuple* $(N, \Theta, P, \{A_{i,\theta_i}\}_{i \in N, \theta_i \in \Theta_i},$ $G, \{u^\alpha\}_{\alpha \in \mathcal{A}})$ *where* $N$ *is the set of agents;* $\Theta = \prod_i \Theta_i$ *is the set of type profiles;* $P$ *is the type distribution, represented as a Bayesian network;* $A_{i,\theta_i} \subseteq \mathcal{A}$ *is the type-action set of* $i$ *given* $\theta_i$; $G = (\mathcal{A}, E)$ *is the action graph; and for each* $\alpha \in \mathcal{A}$, *the utility function is* $u^\alpha : C^{(\alpha)} \to \mathbb{R}$.

Intuitively, this representation captures two types of structure in utility functions: firstly, shared actions capture the game's *anonymity* structure: if two action choices from different type-action sets share an action node $\alpha$, it means that these two actions are interchangeable as far as the other players' utilities are concerned. In other words, their utilities may depend on the *number* of players that chose the action node $\alpha$, but not the identities of those players. Secondly, the (lack of) edges between nodes in the action graph expresses *action- and type-specific independencies* of utilities of the game: depending on player $i$'s chosen action node (which also encodes information about her type), her utility depends on configurations over different sets of nodes.

**Lemma 4.** *An arbitrary Bayesian game given in Bayesian normal form can be encoded as a BAGG storing the same number of utility values.*

*Proof.* Provided in the supplementary material. $\square$

Bayesian games with symmetric utility functions exhibit anonymity structure, which can be expressed in BAGGs by sharing action nodes. Specifically, we label each $\Theta_i$ as $\{1, \ldots, T\}$, so that each $t \in \{1, \ldots, T\}$ corresponds to a class of equivalent types. Then for each $t \in \{1, \ldots, T\}$, we have $A_{i,t} = A_{j,t}$ for all $i, j \in N$, i.e. type-action sets for equivalent types are identical.

### 3.1 BAGGs with function nodes

In this section we extend the basic BAGG representation by introducing *function nodes* to the action graph. The concept of function nodes was first introduced in the (complete-information) AGG setting [13]. Function nodes allow us to exploit a much wider variety of utility structures in BAGGs.

In this extended representation, the action graph $G$'s vertices consist of both the set of action nodes $\mathcal{A}$ and the set of function nodes $\mathcal{F}$. We require that no function node $p \in \mathcal{F}$ can be in any player's action set. Each function node $p \in \mathcal{F}$ is associated with a function $f^p : C^{(p)} \to \mathbb{R}$. We extend $c$ by defining $c(p)$ to be the result of applying $f^p$ to the configuration over $p$'s neighbors, $f^p(c^{(p)})$. Intuitively, $c(p)$ can be used to describe intermediate parameters that players' utilities depend on. To ensure that the BAGG is meaningful, the graph restricted to nodes in $\mathcal{F}$ is required to be a directed acyclic graph. As before, for each action node $\alpha$ we define a utility function $u^\alpha : C^{(\alpha)} \to \mathbb{R}$.

Of particular computational interest is the subclass of *contribution-independent function nodes* (also introduced by [13]). A function node $p$ in a BAGG is *contribution-independent* if $\nu(p) \subseteq \mathcal{A}$, there exists a commutative and associative operator $*$, and for each $\alpha \in \nu(p)$ an integer $w_\alpha$, such that given an action profile $a = (a_1, \ldots, a_n)$, $c(p) = *_{i \in N : a_i \in \nu(p)} w_{a_i}$. A BAGG is contribution-independent if all its function nodes are contribution-independent. Intuitively, if function node $p$ is contribution-independent, each player's strategy affects $c(p)$ independently.

A very useful kind of contribution-independent function nodes are *counting function nodes*, which set $*$ to the summation operator $+$ and the weights to 1. Such a function node $p$ simply counts the number of players that chose any action in $\nu(p)$.

Let us consider the size of a BAGG representation. The representation size of the Bayesian network for $P$ is exponential only in the in-degree of the BN. The utility functions store $\sum_\alpha |C^{(\alpha)}|$ values. As in similar analysis for AGGs [15], estimations of this size generally depend on what types of function nodes are included. We state only the following (relatively straightforward) result since in this paper we are mostly concerned with BAGGs with counting function nodes.

**Theorem 5.** *Consider BAGGs whose only function nodes, if any, are counting function nodes. If the in-degrees of the action nodes as well as the in-degrees of the Bayesian networks for* $P$ *are bounded by a constant, then the sizes of the BAGGs are bounded by a polynomial in* $n$, $|\mathcal{A}|$, $|\mathcal{F}|$, $\sum_i |\Theta_i|$ *and the sizes of domains of variables in the BN.*

This theorem shows a nice property of counting function nodes: representation size does not grow exponentially in the in-degrees of these counting function nodes. The next example illustrates the usefulness of counting function nodes, including for expressing conditional utility independence.

**Example 6** (Coffee Shop game). *Consider a symmetric Bayesian game involving $n$ players; each player plans to open a new coffee shop in a downtown area, but has to decide on the location. The downtown area is represented by a $r \times k$ grid. Each player can choose to open a shop located within any of the $B \equiv rk$ blocks or decide not to enter the market. Each player has $T$ types, representing her private information about her cost of opening a coffee shop. Players' types are independently distributed. Conditioned on player $i$ choosing some location, her utility depends on: (a) her own type; (b) the number of players that chose the same block; (c) the number of players that chose any of the surrounding blocks; and (d) the number of players that chose any other location.*

The Bayesian normal form representation of this game has size $n[T(B+1)]^n$. The game can be expressed as a BAGG as follows. Since the game is symmetric, we label the types as $\{1, \ldots, T\}$. $\mathcal{A}$ contains one action $O$ corresponding to not entering and $TB$ other action nodes, with each location corresponding to a set of $T$ action nodes, each representing the choice of that location by a player with a different type. For each $t \in \{1, \ldots, T\}$, the type-action sets $A_{i,t} = A_{j,t}$ for all $i, j \in N$ and each consists of the action $O$ and $B$ actions corresponding to locations for type $t$. For each location $(x, y)$ we create three function nodes: $p_{xy}$ representing the number of players choosing this location, $p'_{xy}$ representing the number of players choosing any surrounding blocks, and $p''_{xy}$ representing the number of players choosing any other block. Each of these function nodes is a counting function node, whose neighbors are action nodes corresponding to the appropriate locations (for all types). Each action node for location $(x, y)$ has three neighbors, $p_{xy}$, $p'_{xy}$, and $p''_{xy}$. Since the BAGG action graph has maximum in-degree 3, by Theorem 5 the representation size is polynomial in $n$, $B$ and $T$.

## 4 Computing a Bayes-Nash Equilibrium

In this section we consider the problem of finding a sample Bayes-Nash equilibrium given a BAGG. Our overall approach is to interpret the Bayesian game as a complete-information game, and then to apply existing algorithms for finding Nash equilibria of complete-information games. We consider two state-of-the-art Nash equilibrium algorithms, van der Laan et al's simplicial subdivision [24] and Govindan and Wilson's global Newton method [9]. Both run in exponential time in the worst case, and indeed recent complexity theoretic results [3, 6, 4] imply that a polynomial-time algorithm for Nash equilibrium is unlikely to exist.[1] Nevertheless, we show that we can achieve exponential speedups in these algorithms by exploiting the structure of BAGGs.

Recall from Section 2.2.1 that a Bayesian game can be transformed into its induced normal form or its agent form. In the induced normal form, each player $i$ has $|A_i|^{|\Theta_i|}$ actions (corresponding to her pure strategies of the Bayesian game). Solving such a game would be infeasible for large $|\Theta_i|$; just to represent an Nash equilibrium requires space exponential in $|\Theta_i|$.

A more promising approach is to consider the agent form. Note that we can straightforwardly adapt the agent-form transformation described in Section 2.2.1 to the setting of BAGGs: now the action set of player $(i, \theta_i)$ of the agent form corresponds to the type-action set $A_{i,\theta_i}$ of the BAGG. The resulting complete-information game has $\sum_{i \in N} |\Theta_i|$ players and $|A_{i,\theta_i}|$ actions for each player $(i, \theta_i)$; a Nash equilibrium can be represented using just $\sum_i \sum_{\theta_i} |A_{i,\theta_i}|$ numbers. However, the normal form representation of the agent form has size $\sum_{j \in N} |\Theta_j| \prod_{i,\theta_i} |A_{i,\theta_i}|$, which grows exponentially in $n$ and $|\Theta_i|$. Applying the Nash equilibrium algorithms to this normal form would be infeasible in terms of time and space. Fortunately, we do not have to explicitly represent the agent form as a normal form game. Instead, we treat a BAGG as a compact representation of its agent form, and carry out any required computation on the agent form by operating on the BAGG. A key computational task required by both Nash equilibrium algorithms in their inner loops is the computation of expected utility of the agent form. Recall from Section 2.2.1 that for all $(i, \theta_i)$ the expected utility $\tilde{u}_{i,\theta_i}(\tilde{\sigma})$ of the agent form is equal to the expected utility $u_i(\sigma | \theta_i)$ of the Bayesian game. Thus in the remainder of this section we focus on the problem of computing expected utility in BAGGs.

### 4.1 Computing Expected Utility in BAGGs

Recall that $\sigma^{\theta_i \to a_i}$ is the mixed strategy profile that is identical to $\sigma$ except that $i$ plays $a_i$ given $\theta_i$. The main quantity we are interested in is $u_i(\sigma^{\theta_i \to a_i} | \theta_i)$, player $i$'s expected utility given $\theta_i$ under

the strategy profile $\sigma^{\theta_i \to a_i}$. Note that the expected utility $u_i(\sigma|\theta_i)$ can then be computed as the sum $u_i(\sigma|\theta_i) = \sum_{a_i} u_i(\sigma^{\theta_i \to a_i}|\theta_i)\sigma_i(a_i|\theta_i)$.

One approach is to directly apply Equation (1), which has $(|\Theta_{-i}| \times |A|)$ terms in the summation. For games represented in Bayesian normal form, this algorithm runs in time polynomial in the representation size. Since BAGGs can be exponentially more compact than their equivalent Bayesian normal form representations, this algorithm runs in exponential time for BAGGs.

In this section we present a more efficient algorithm that exploits BAGG structure. We first formulate the expected utility problem as a Bayesian network inference problem. Given a BAGG and a mixed strategy profile $\sigma^{\theta_i \to a_i}$, we construct the *induced Bayesian network (IBN)* as follows.

We start with the BN representing the type distribution $P$, which includes (at least) the random variables $\theta_1, \ldots, \theta_n$. The conditional probability distributions (CPDs) for the network are unchanged. We add the following random variables: one strategy variable $D_j$ for each player $j$; one action count variable for each action node $\alpha \in \mathcal{A}$, representing its action count, denoted $c(\alpha)$; one function variable for each function node $p \in \mathcal{F}$, representing its configuration value, denoted $c(p)$; and one utility variable $U^\alpha$ for each action node $\alpha$. We then add the following edges: an edge from $\theta_j$ to $D_j$ for each player $j$; for each player $j$ and each $\alpha \in A_j^\cup$, an edge from $D_j$ to $c(\alpha)$; for each function variable $c(p)$, all incoming edges corresponding to those in the action graph $G$; and for each $\alpha \in \mathcal{A}$, for each action or function node $m \in \nu(\alpha)$ in $G$, an edge from $c(m)$ to $U^\alpha$ in the IBN.

The CPDs of the newly added random variables are defined as follows. Each strategy variable $D_j$ has domain $A_j^\cup$, and given its parent $\theta_j$, its CPD chooses an action from $A_j^\cup$ according to the mixed strategy $\sigma_j^{\theta_i \to a_i}$. In other words, if $j \neq i$ then $\Pr(D_j = a_j|\theta_j)$ is equal to $\sigma_j(a_j|\theta_j)$ for all $a_j \in A_{j,\theta_j}$ and 0 for all $a_j \in A_j^\cup \setminus A_{j,\theta_j}$; and if $j = i$ we have $\Pr(D_j = a_i|\theta_j) = 1$. For each action node $\alpha$, the parents of its action-count variable $c(\alpha)$ are strategy variables that have $\alpha$ in their domains. The CPD is a deterministic function that returns the number of its parents that take value $\alpha$; i.e., it calculates the action count of $\alpha$. For each function variable $c(p)$, its CPD is the deterministic function $f^p$. The CPD for each utility variable $U^\alpha$ is a deterministic function specified by $u^\alpha$.

It is straightforward to verify that the IBN is a directed acyclic graph (DAG) and thus represents a valid joint distribution. Furthermore, the expected utility $u_i(\sigma^{t_i \to a_i}|\theta_i)$ is exactly the expected value of the variable $U^{a_i}$ conditioned on the instantiated type $\theta_i$.

**Lemma 7.** *For all $i \in N$, all $\theta_i \in \Theta_i$ and all $a_i \in A_{i,\theta_i}$, we have $u_i(\sigma^{\theta_i \to a_i}|\theta_i) = E[U^{a_i}|\theta_i]$.*

Standard BN inference methods could be used to compute $E[U^{a_i}|\theta_i]$. However, such standard algorithms do not take advantage of structure that is inherent in BAGGs. In particular, recall that in the induced network, each action count variable $c(\alpha)$'s parents are all strategy variables that have $\alpha$ in their domains, implying large in-degrees for action count variables. Applying (e.g.) the clique-tree algorithm would yield large clique sizes, which is problematic because running time scales exponentially in the largest clique size of the clique tree. However, the CPDs of these action count variables are structured counting functions. Such structure is an instance of *causal independence* in BNs [11]. It also corresponds to anonymity structure for complete-information game representations like symmetric games and AGGs [13]. We can exploit this structure to speed up computation of expected utility in BAGGs. Our approach is a specialization of Heckerman and Breese's method [11] for exploiting causal independence in BNs, which transforms the original BN by creating new nodes that represent intermediate results, and re-wiring some of the arcs, resulting in an equivalent BN with small in-degree. Given an action count variable $c(\alpha)$ with parents (say) $\{D_1 \ldots D_n\}$, for each $i \in \{1 \ldots n-1\}$ we create a node $M_{\alpha,i}$, representing the count induced by $D_1 \ldots D_i$. Then, instead of having $D_1 \ldots D_n$ as parents of $c(\alpha)$, its parents become $D_n$ and $M_{\alpha,n-1}$, and each $M_{\alpha,i}$'s parents are $D_i$ and $M_{\alpha,i-1}$. The resulting graph has in-degree at most 2 for $c(\alpha)$ and the $M_{\alpha,i}$'s. The CPDs of function variables corresponding to contribution-independent function nodes also exhibit causal independence, and thus we can use a similar transformation to reduce their in-degree to 2. We call the resulting Bayesian network the *transformed Bayesian network (TBN)* of the BAGG.

It is straightforward to verify that the representation size of the TBN is polynomial in the size of the BAGG. We can then use standard inference algorithms to compute $E[U^\alpha|\theta_i]$ on the TBN. For classes of BNs with bounded treewidths, this can be computed in polynomial time. Since the graph structure (and thus the treewidth) of the TBN does not depend on the strategy profile and only depends on the BAGG, we have the following result.

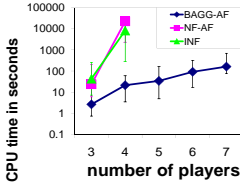
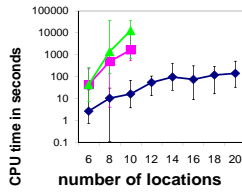
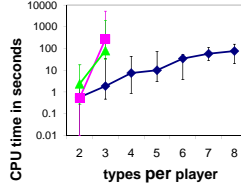
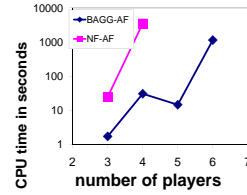

Figure 1: GW, varying players.

Figure 2: GW, varying locations.

Figure 3: GW, varying types.

Figure 4: simplicial subdivision.

**Theorem 8.** *For BAGGs whose TBNs have bounded treewidths, expected utility can be computed in time polynomial in $n$, $|\mathcal{A}|$, $|\mathcal{F}|$ and $|\sum_i \Theta_i|$.*

Bayesian games with independent type distributions are an important class of games and have many applications, such as independent-private-value auctions. When contribution-independent BAGGs have independent type distributions, expected utility can be efficiently computed.

**Theorem 9.** *For contribution-independent BAGGs with independent type distributions, expected utility can be computed in time polynomial in the size of the BAGG.*

*Proof.* Provided in the supplementary material. □

Note that this result is stronger than that of Theorem 8, which only guarantees efficient computation when TBNs have constant treewidth.

## 5 Experiments

We have implemented our approach for computing a Bayes-Nash equilibrium given a BAGG by applying Nash equilibrium algorithms on the agent form of the BAGG. We adapted two algorithms, GAMBIT's [18] implementation of simplicial subdivision and GameTracer's [2] implementation of Govindan and Wilson's global Newton method, by replacing calls to expected utility computations of the complete-information game with corresponding expected utility computations of the BAGG. We ran experiments that tested the performance of our approach (denoted by BAGG-AF) against two approaches that compute a Bayes-Nash equilibrium for arbitrary Bayesian games. The first (denoted INF) computes a Nash equilibrium on the induced normal form; the second (denoted NF-AF) computes a Nash equilibrium on the normal form representation of the agent form. Both were implemented using the original, normal-form-based implementations of simplicial subdivision and global Newton method. We thus studied six concrete algorithms, two for each game representation.

We tested these algorithms on instances of the Coffee Shop Bayesian game described in Example 6. We created games of different sizes by varying the number of players, the number of types per player and the number of locations. For each size we generated 10 game instances with random integer payoffs, and measured the running (CPU) times. Each run was cut off after 10 hours if it had not yet finished. All our experiments were performed using a computer cluster consisting of 55 machines with dual Intel Xeon 3.2GHz CPUs, 2MB cache and 2GB RAM, running Suse Linux 11.1.

We first tested the three approaches based on the Govindan-Wilson (GW) algorithm. Figure 1 shows running time results for Coffee Shop games with $n$ players, 2 types per player on a $2 \times 3$ grid, with $n$ varying from 3 to 7. Figure 2 shows running time results for Coffee Shop games with 3 players, 2 types per player on a $2 \times x$ grid, with $x$ varying from 3 to 10. Figure 3 shows results for Coffee Shop games with 3 players, $T$ types per player on a $1 \times 3$ grid, with $T$ varying from 2 to 8. The data points represent the median running time of 10 game instances, with the error bars indicating the maximum and minimum running times. All results show that our BAGG-based approach (BAGG-AF) significantly outperformed the two normal-form-based approaches (INF and NF-AF). Furthermore, as we increased the dimensions of the games the normal-form based approaches quickly ran out of memory (hence the missing data points), whereas BAGG-NF did not.

We also did some preliminary experiments on BAGG-AF and NF-AF running the simplicial subdivision algorithm. Figure 4 shows running time results for Coffee Shop games with $n$ players, 2 types per player on a $1 \times 3$ grid, with $n$ varying from 3 to 6. Again, BAGG-AF significantly outperformed NF-AF, and NF-AF ran out of memory for game instances with more than 4 players.

## Footnotes

[1] There has been some research on efficient Nash-equilibrium-finding algorithms for subclasses of games, such as Daskalakis and Papadimitriou's [5] PTAS for anonymous games with fixed numbers of actions. One future direction would be to adapt these algorithms to subclasses of Bayesian games.

# References

[1] N. Bhat and K. Leyton-Brown. Computing Nash equilibria of action-graph games. In *UAI*, pages 35–42, 2004.

[2] B. Blum, C. Shelton, and D. Koller. Gametracer. `http://dags.stanford.edu/Games/gametracer.html`, 2002.

[3] X. Chen and X. Deng. Settling the complexity of 2-player Nash-equilibrium. In *FOCS: Proceedings of the Annual IEEE Symposium on Foundations of Computer Science*, pages 261–272, 2006.

[4] C. Daskalakis, P. W. Goldberg, and C. H. Papadimitriou. The complexity of computing a Nash equilibrium. In *STOC: Proceedings of the Annual ACM Symposium on Theory of Computing*, pages 71–78, 2006.

[5] C. Daskalakis and C. Papadimitriou. Computing equilibria in anonymous games. In *FOCS: Proceedings of the Annual IEEE Symposium on Foundations of Computer Science*, pages 83–93, 2007.

[6] P. W. Goldberg and C. H. Papadimitriou. Reducibility among equilibrium problems. In *STOC: Proceedings of the Annual ACM Symposium on Theory of Computing*, pages 61–70, 2006.

[7] G. Gottlob, G. Greco, and T. Mancini. Complexity of pure equilibria in Bayesian games. In *IJCAI*, pages 1294–1299, 2007.

[8] S. Govindan and R. Wilson. Structure theorems for game trees. *Proceedings of the National Academy of Sciences*, 99(13):9077–9080, 2002.

[9] S. Govindan and R. Wilson. A global Newton method to compute Nash equilibria. *Journal of Economic Theory*, 110:65–86, 2003.

[10] J.C. Harsanyi. Games with incomplete information played by "Bayesian" players, i-iii. part i. the basic model. *Management science*, 14(3):159–182, 1967.

[11] David Heckerman and John S. Breese. Causal independence for probability assessment and inference using Bayesian networks. *IEEE Transactions on Systems, Man and Cybernetics*, 26(6):826–831, 1996.

[12] J.T. Howson Jr and R.W. Rosenthal. Bayesian equilibria of finite two-person games with incomplete information. *Management Science*, pages 313–315, 1974.

[13] A. X. Jiang and K. Leyton-Brown. A polynomial-time algorithm for Action-Graph Games. In *AAAI*, pages 679–684, 2006.

[14] A. X. Jiang, A. Pfeffer, and K. Leyton-Brown. Temporal Action-Graph Games: A new representation for dynamic games. In *UAI*, 2009.

[15] Albert Xin Jiang, Kevin Leyton-Brown, and Navin Bhat. Action-graph games. *Games and Economic Behavior*, 2010. In press.

[16] M.J. Kearns, M.L. Littman, and S.P. Singh. Graphical models for game theory. In *UAI*, pages 253–260, 2001.

[17] D. Koller and B. Milch. Multi-agent influence diagrams for representing and solving games. In *IJCAI*, 2001.

[18] R. D. McKelvey, A. M. McLennan, and T. L. Turocy. Gambit: Software tools for game theory, 2006. `http://econweb.tamu.edu/gambit`.

[19] N. Nisan, T. Roughgarden, E. Tardos, and V. Vazirani, editors. *Algorithmic Game Theory*. Cambridge University Press, Cambridge, UK, 2007.

[20] Frans A. Oliehoek, Matthijs T. J. Spaan, Jilles Dibangoye, and Christopher Amato. Heuristic search for identical payoff bayesian games. In *AAMAS: Proceedings of the International Joint Conference on Autonomous Agents and Multiagent Systems*, pages 1115–1122, May 2010.

[21] Daniel M. Reeves and Michael P. Wellman. Computing best-response strategies in infinite games of incomplete information. In *UAI*, pages 470–478, 2004.

[22] Y. Shoham and K. Leyton-Brown. *Multiagent Systems: Algorithmic, Game-Theoretic, and Logical Foundations*. Cambridge University Press, New York, 2009.

[23] S. Singh, V. Soni, and M. Wellman. Computing approximate Bayes-Nash equilibria in tree-games of incomplete information. In *EC: Proceedings of the ACM Conference on Electronic Commerce*, pages 81–90. ACM, 2004.

[24] G. van der Laan, A.J.J. Talman, and L. van der Heyden. Simplicial variable dimension algorithms for solving the nonlinear complementarity problem on a product of unit simplices using a general labelling. *Mathematics of Operations Research*, 12(3):377–397, 1987.

[25] Yevgeniy Vorobeychik. *Mechanism Design and Analysis Using Simulation-Based Game Models*. PhD thesis, University of Michigan, 2008.

